# Multi-time Models for Temporally Abstract Planning

**Doina Precup, Richard S. Sutton**
University of Massachusetts
Amherst, MA 01003
{dprecup|rich}@cs.umass.edu

## Abstract

Planning and learning at multiple levels of temporal abstraction is a key problem for artificial intelligence. In this paper we summarize an approach to this problem based on the mathematical framework of Markov decision processes and reinforcement learning. Current model-based reinforcement learning is based on one-step models that cannot represent common-sense higher-level actions, such as going to lunch, grasping an object, or flying to Denver. This paper generalizes prior work on temporally abstract models [Sutton, 1995] and extends it from the prediction setting to include actions, control, and planning. We introduce a more general form of temporally abstract model, the *multi-time model*, and establish its suitability for planning and learning by virtue of its relationship to the Bellman equations. This paper summarizes the theoretical framework of multi-time models and illustrates their potential advantages in a gridworld planning task.

The need for hierarchical and abstract planning is a fundamental problem in AI (see, e.g., Sacerdoti, 1977; Laird et al., 1986; Korf, 1985; Kaelbling, 1993; Dayan & Hinton, 1993). Model-based reinforcement learning offers a possible solution to the problem of integrating planning with real-time learning and decision-making (Peng & Williams, 1993, Moore & Atkeson, 1993; Sutton and Barto, 1998). However, current model-based reinforcement learning is based on one-step models that cannot represent common-sense, higher-level actions. Modeling such actions requires the ability to handle different, interrelated levels of temporal abstraction.

A new approach to modeling at multiple time scales was introduced by Sutton (1995) based on prior work by Singh , Dayan , and Sutton and Pinette . This approach enables models of the environment at different temporal scales to be intermixed, producing temporally abstract models. However, that work was concerned only with predicting the environment. This paper summarizes an extension of the approach including actions and control of the environment [Precup & Sutton, 1997]. In particular, we generalize the usual notion of a

primitive, one-step action to an *abstract action*, an arbitrary, closed-loop policy. Whereas prior work modeled the behavior of the agent-environment system under a single, given policy, here we learn different models for a set of different policies. For each possible way of behaving, the agent learns a separate model of what will happen. Then, in planning, it can choose between these overall policies as well as between primitive actions.

To illustrate the kind of advance we are trying to make, consider the example shown in Figure 1. This is a standard gridworld in which the primitive actions are to move from one grid cell to a neighboring cell. Imagine the learning agent is repeatedly given new tasks in the form of new goal locations to travel to as rapidly as possible. If the agent plans at the level of primitive actions, then its plans will be many actions long and take a relatively long time to compute. Planning could be much faster if abstract actions could be used to plan for moving from room to room rather than from cell to cell. For each room, the agent learns two models for two abstract actions, one for traveling efficiently to each adjacent room. We do not address in this paper the question of how such abstract actions could be discovered without help; instead we focus on the mathematical theory of abstract actions. In particular, we define a very general semantics for them—a property that seems to be required in order for them to be used in the general kind of planning typically used with Markov decision processes. At the end of this paper we illustrate the theory in this example problem, showing how room-to-room abstract actions can substantially speed planning.

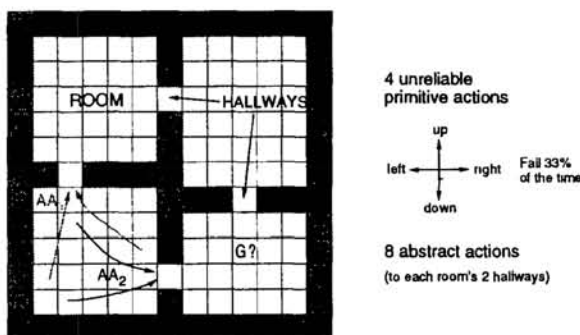

Figure 1: Example Task. The Natural abstract actions are to move from room to room.

# 1    Reinforcement Learning (MDP) Framework

In reinforcement learning, a learning *agent* interacts with an *environment* at some discrete, lowest-level time scale $t = 0, 1, 2, \ldots$ On each time step, the agent perceives the state of the environment, $s_t$, and on that basis chooses a primitive action, $a_t$. In response to each primitive action, $a_t$, the environment produces one step later a numerical reward, $r_{t+1}$, and a next state, $s_{t+1}$. The agent's objective is to learn a policy, a mapping from states to probabilities of taking each action, that maximizes the expected discounted future reward from each state $s$:

$$v^{\pi}(s) = E_{\pi}\left\{ \sum_{t=0}^{\infty} \gamma^t r_{t+1} \,\middle|\, s_0 = s \right\},$$

where $\gamma \in [0, 1)$ is a *discount-rate* parameter, and $E_{\pi}\{\}$ denotes an expectation implicitly conditional on the policy $\pi$ being followed. The quantity $v^{\pi}(s)$ is called the *value* of state $s$ under policy $\pi$, and $v^{\pi}$ is called the value function for policy $\pi$. The value under the optimal policy is denoted:

$$v^*(s) = \max_{\pi} v^{\pi}(s).$$

Planning in reinforcement learning refers to the use of models of the effects of actions to compute value functions, particularly $v^*$.

We assume that the states are discrete and form a finite set, $s_t \in \{1, 2, \ldots, m\}$. This is viewed as a temporary theoretical convenience; it is not a limitation of the ideas we present. This assumption allows us to alternatively denote the value functions, $v^\pi$ and $v^*$, as column vectors, $\mathbf{v}^\pi$ and $\mathbf{v}^*$, each having $m$ components that contain the values of the $m$ states. In general, for any $m$-vector, $\mathbf{x}$, we will use the notation $x(s)$ to refer to its $s$th component.

The model of an action, $a$, whether primitive or abstract, has two components. One is an $m \times m$ matrix, $P_a$, predicting the state that will result from executing the action in each state. The other is a vector, $\mathbf{g}_a$, predicting the cumulative reward that will be received along the way. In the case of a primitive action, $P_a$ is the matrix of 1-step transition probabilities of the environment, times $\gamma$:

$$P_a^T(s) = \gamma E \{\mathbf{s}_{t+1} \mid s_t = s, a_t = a\}, \qquad \forall s$$

where $P_a^T(s)$ denotes the $s$th column of $P_a^T$ (these are the predictions corresponding to state $s$) and $\mathbf{s}_t$ denotes the unit basis $m$-vector corresponding to $s_t$. The reward prediction, $\mathbf{g}_a$, for a primitive action contains the expected immediate rewards:

$$g_a(s) = E \{r_{t+1} \mid s_t = s, a_t = a\}, \qquad \forall s$$

For any stochastic policy, $\pi$, we can similarly define its 1-step model, $\mathbf{g}_\pi, P_\pi$ as:

$$P_\pi^T(s) = \gamma E_\pi \Big\{ \mathbf{s}_{t+1} \Big| s_t = s \Big\} \qquad \text{and} \qquad g_\pi(s) = E_\pi \Big\{ r_{t+1} \Big| s_t = s \Big\} \qquad \forall s \quad (1)$$

## 2   Suitability for Planning

In conventional planning, one-step models are used to compute value functions via the Bellman equations for prediction and control. In vector notation, the prediction and control Bellman equations are

$$\mathbf{v}^\pi = \mathbf{g}_\pi + P_\pi \mathbf{v}^\pi \qquad \text{and} \qquad \mathbf{v}^* = \max_a \{\mathbf{g}_a + P_a \mathbf{v}^*\}, \qquad (2)$$

respectively, where the max function is applied component-wise in the control equation. In planning, these equalities are turned into updates, e.g., $\mathbf{v}_{k+1}^\pi \leftarrow \mathbf{g}_\pi + P_\pi \mathbf{v}_k^\pi$, which converge to the value functions. Thus, the Bellman equations are usually used to define and compute value functions given models of actions. Following Sutton (1995), here we reverse the roles: we take the value functions as given and use the Bellman equations to define and compute models of new, abstract actions.

In particular, a model can be used in planning only if it is stable and consistent with the Bellman equations. It is useful to define special terms for consistency with each Bellman equation. Let $\mathbf{g}, P$ denote an arbitrary model (an $m$-vector and an $m \times m$ matrix). Then this model is said to be *valid* for policy $\pi$ [Sutton, 1995] if and only if $\lim_{k \to \infty} P^k = 0$ and

$$\mathbf{v}^\pi = \mathbf{g} + P\mathbf{v}^\pi. \qquad (3)$$

Any valid model can be used to compute $\mathbf{v}^\pi$ via the iteration algorithm $\mathbf{v}_{k+1}^\pi \leftarrow \mathbf{g} + P\mathbf{v}_k^\pi$. This is a direct sense in which the validity of a model implies that it is suitable for planning. We introduce here a parallel definition that expresses consistency with the control Bellman equation. The model $\mathbf{g}, P$ is said to be *non-overpromising* (NOP) if and only if $P$ has only positive elements, $\lim_{k \to \infty} P^k = 0$, and

$$\mathbf{v}^* \geq \mathbf{g} + P\mathbf{v}^*, \qquad (4)$$

where the $\geq$ relation holds component-wise. If a NOP model is added inside the max operator in the control Bellman equation (2), this condition ensures that the true value, $\mathbf{v}^*$, will not be exceeded for any state. Thus, any model that does not promise more than it

is achievable (is not overpromising) can serve as an option for planning purposes. The one-step models of primitive actions are obviously NOP, due to (2). It is similarly straightforward to show that the one-step model of any policy is also NOP.

For some purposes, it is more convenient to write a model $g$, $P$ as a single $(m+1) \times (m+1)$ matrix:

$$M = \left[ \begin{array}{c|c} 1 & - \quad 0 \quad - \\ \hline g & P \end{array} \right].$$

We say that the model $M$ has been put in homogeneous coordinates. The vectors corresponding to the value functions can also be put into homogeneous coordinates, by adding an initial element that is always 1.

Using this notation, new models can be combined using two basic operations: composition and averaging. Two models $M_1$ and $M_2$ can be *composed* by matrix multiplication, yielding a new model $M = M_1 M_2$. A set of models $M_i$ can be *averaged*, weighted by a set of diagonal matrices $D_i$, such that $\sum_i D_i = I$, to yield a new model $M = \sum_i D_i M_i$. Sutton (1995) showed that the set of models that are valid for a policy $\pi$ is closed under composition and averaging. This enables models acting at different time scales to be mixed together, and the resulting model can still be used to compute $v^\pi$. We have proven that the set of NOP models is also closed under composition and averaging [Precup & Sutton, 1997]. These operations permit a richer variety of combinations for NOP models than they do for valid models because the NOP models that are combined need not correspond to a particular policy.

## 3 Multi-time models

The validity and NOP-ness of a model do not imply each other [Precup & Sutton, 1997]. Nevertheless, we believe a good model should be both valid and NOP. We would like to describe a class of models that, in some sense, includes all the "interesting" models that are valid and non-overpromising, and which is expressive enough to include common-sense notions of abstract action. These goals have led us to the notion of a *multi-time model*.

The simplest example of multi-step model, called the *n-step model for policy $\pi$*, predicts the $n$-step truncated return and the state $n$ steps into the future (times $\gamma^n$). If different $n$-step models of the same policy are averaged, the result is called a *mixture model*. Mixtures are valid and non-overpromising due to the closure properties established in the previous section. One kind of mixture suggested in [Sutton, 1995] allows an exponential decay of the weights over time, controlled by a parameter $\beta$.

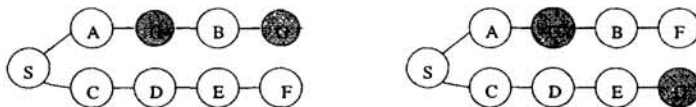

Figure 2: Two hypothetical Markov environments

Are mixture models expressive enough for capturing the properties of the environment? In order to get some intuition about the expressive power that a model should have, let us consider the example in figure 2. If we are only interested if state G is attained, then the two environments presented should be characterized by significantly different models. However, $n$-step models, or any linear mixture of $n$-step models cannot achieve this goal. In order to remediate this problem, models should average differently over all the different trajectories that are possible through the state space. A full $\beta$-model [Sutton, 1995] can

distinguish between these two situations. A $\beta$-model is a more general form of mixture model, in which a different $\beta$ parameter is associated with each state. For a state $i$, $\beta_i$ can be viewed as the probability that the trajectory through the state space ends in state $i$. Although $\beta$-models seem to have more expressive power, they cannot describe $n$-step models. We would like to have a more general form of model, that unifies both classes. This goal is achieved by accurate multi-time models.

Multi-time models are defined with respect to a policy. Just as the one-step model for a policy is defined by (1), we define $\mathbf{g}, P$ to be an accurate multi-time model if and only if

$$P^T(s) = E_\pi\Big\{\sum_{t=1}^{\infty} w_t\, \gamma^t\, \mathbf{s}_t \;\Big|\; s_0 = s\Big\},$$

$$g(s) = E_\pi\Big\{\sum_{t=1}^{\infty} w_t\, (r_1 + \gamma r_2 + \cdots + \gamma^{t-1} r_t) \;\Big|\; s_0 = s\Big\}$$

for some $\pi$, for all $s$, and for some sequence of random weights, $w_1, w_2, \ldots$ such that $w_t > 0$ and $\sum_{t=1}^{\infty} w_t = 1$. The weights are random variables chosen according to a distribution that depends only on states visited at or before time $t$. The weight $w_t$ is a measure of the importance given to the $t$-th state of the trajectory. In particular, if $w_t = 0$, then state $t$ has no weight associated with it. If $w_t = 1 - \sum_{i=0}^{t-1} w_i$, all the remaining weight along the trajectory is given to state $t$. The effect is that state $\dot{s}_t$ is the "outcome" state for the trajectory.

The random weights along each trajectory make this a very general form of model. The only necessary constraint is that the weights depend only on previously visited states. In particular, we can choose weighting sequences that generate the types of multi-step models described in [Sutton, 1995]. If the weighting variables are such that $w_n{=}1$, and $w_t = 0, \forall t \neq n$, we obtain $n$-step models. A weighting sequence of the form $w_t = \Pi_{i=0}^{t-1}\beta_i \,\forall t$, where $\beta_i$ is the parameter associated to the state visited on time step $i$, describes a full $\beta$-model.

The main result for multi-time models is that they satisfy the two criteria defined in the previous section. Any accurate multi-time model is also NOP and valid for $\pi$. The proofs of these results are too long to include here.

## 4   Illustrative Example

In order to illustrate the way in which multi-time models can be used in practice, let us return to the gridworld example (Figure 1). The cells of the grid correspond to the states of the environment. From any state the agent can perform one of four primitive actions, up, down, left or right. With probability 2/3, the actions cause the agent to move one cell in the corresponding direction (unless this would take the agent into a wall, in which case it stays in the same state). With probability 1/3, the agent instead moves in one of the other three directions (unless this takes it into a wall of course). There is no penalty for bumping into walls.

In each room, we also defined two abstract actions, for going to each of the adjacent hallways. Each abstract action has a set of input states (the states in the room) and two outcome states: the target hallway, which corresponds to a successful outcome, and the state adjacent to the other hallway, which corresponds to failure (the agent has wandered out of the room). Each abstract action is given by its complete model $\mathbf{g}_w^\pi, P_w^\pi$, where $\pi$ is the optimal policy for getting into the target hallway, and the weighting variables $w$ along any trajectory have the value 1 for the outcome states and 0 everywhere else.

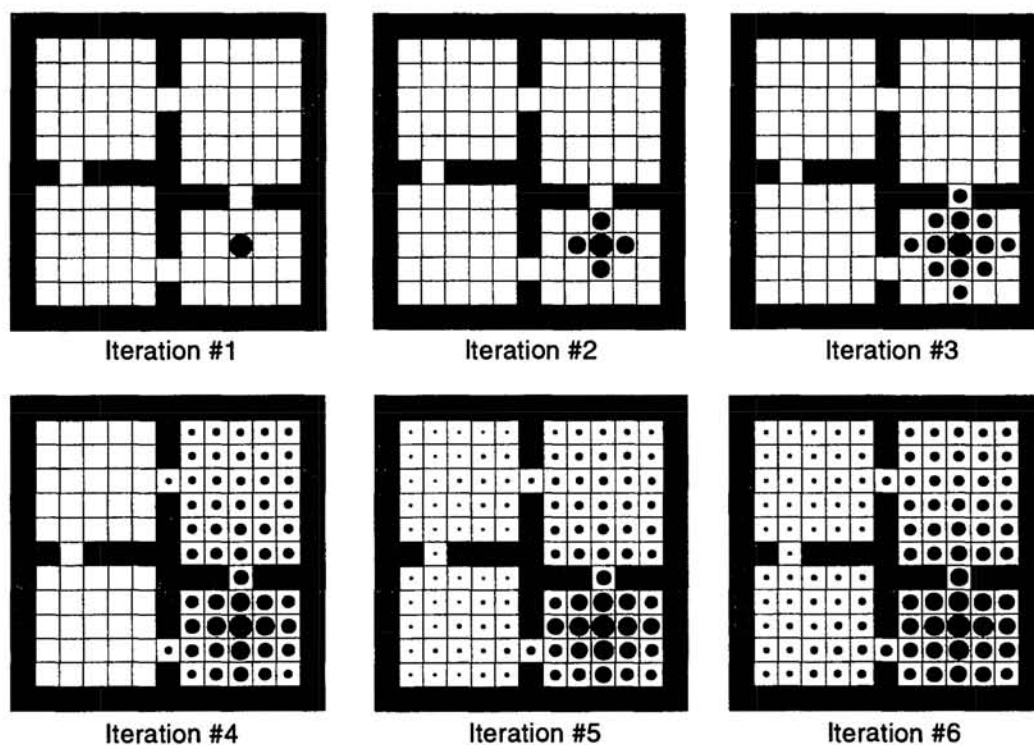

Figure 3: Value iteration using primitive and abstract actions

The goal state can have an arbitrary position in any of the rooms, but for this illustration let us suppose that the goal is two steps down from the right hallway. The value of the goal state is 1, there are no rewards along the way, and the discounting factor is $\gamma = 0.9$. We performed planning according to the standard value iteration method:

$$\mathbf{v}_{k+1} \leftarrow \max_a \mathbf{g}_a + P_a \mathbf{v}_k,$$

where $v_0(s) = 0$ for all the states except the goal state (which starts at 1). In one experiment, $a$ ranged only over the primitive actions, in the other it ranged over the set including both the primitive and the abstract actions.

When using only primitive actions, the values are propagated one step away on each iteration. After six iterations, for instance, only the states that are at most six steps away from the goal will be attributed non-zero values. The models of abstract actions produce a significant speed-up in the propagation of values at each step. Figure 3 shows the value function after each iteration, using both primitive and abstract actions for planning. The area of the circle drawn in each state is proportional to the value attributed to the state. The first three iterations are identical with the case when only primitive actions are used. However, once the values are propagated to the first hallway, all the states in the rooms adjacent to that hallway will receive values as well. For the states in the room containing the goal, these values correspond to performing the abstract action of getting into the right hallway, and then following the optimal primitive actions to get to the goal. At this point, a path to the goal is known from each state in the right half of the environment, even if the path is not optimal for all states. After six iterations, an optimal policy is known for all the states in the environment.

The models of the abstract actions do not need to be given a priori, they can be learned from experience. In fact, the abstract models that were used in this experiment have been learned during a 1,000,000-step random walk in the environment. The starting point for

learning was represented by the outcome states of each abstract action, along with the hypothetical utilities U associated with these states. We used Q-learning [Watkins, 1989] to learn the optimal state-action value function $Q_{U,B}^*$ associated with each abstract action. The greedy policy with respect to $Q_{U,B}^*$ is the policy associated with the abstract action. At the same time, we used the $\beta$-model learning algorithm presented in [Sutton, 1995] to compute the model corresponding to the policy. The learning algorithm is completely online and incremental, and its complexity is comparable to that of regular 1-step TD-learning.

Models of abstract actions can be built while an agent is acting in the environment without any additional effort. Such models can then be used in the planning process as if they would represent primitive actions, ensuring more efficient learning and planning, especially if the goal is changing over time.

## Acknowledgments

The authors thank Amy McGovern and Andy Fagg for helpful discussions and comments contributing to this paper. This research was supported in part by NSF grant ECS-9511805 to Andrew G. Barto and Richard S. Sutton, and by AFOSR grant AFOSR-F49620-96-1-0254 to Andrew G. Barto and Richard S. Sutton. Doina Precup also acknowledges the support of the Fulbright foundation.

# References

Dayan, P. (1993). Improving generalization for temporal difference learning: The successor representation. *Neural Computation, 5*, 613–624.

Dayan, P. & Hinton, G. E. (1993). Feudal reinforcement learning. In *Advances in Neural Information Processing Systems*, volume 5, (pp. 271–278)., San Mateo, CA. Morgan Kaufmann.

Kaelbling, L. P. (1993). Hierarchical learning in stochastic domains: Preliminary results. In *Proceedings of the Tenth International Conference on Machine Learning ICML'93*, (pp. 167–173)., San Mateo, CA. Morgan Kaufmann.

Korf, R. E. (1985). *Learning to Solve Problems by Searching for Macro-Operators*. London: Pitman Publishing Ltd.

Laird, J. E., Rosenbloom, P. S., & Newell, A. (1986). Chunking in SOAR: The anatomy of a general learning mechanism. *Machine Learning, 1*, 11–46.

Moore, A. W. & Atkeson, C. G. (1993). Prioritized sweeping: Reinforcement learning with less data and less real time. *Machine Learning, 13*, 103–130.

Peng, J. & Williams, J. (1993). Efficient learning and planning within the Dyna framework. *Adaptive Behavior, 4*, 323–334.

Precup, D. & Sutton, R. S. (1997). Multi-Time models for reinforcement learning. In *ICML'97 Workshop: The Role of Models in Reinforcement Learning*.

Sacerdoti, E. D. (1977). *A Structure for Plans and Behavior*. North-Holland, NY: Elsevier.

Singh, S. P. (1992). Scaling reinforcement learning by learning variable temporal resolution models. In *Proceedings of the Ninth International Conference on Machine Learning ICML'92*, (pp. 202–207)., San Mateo, CA. Morgan Kaufmann.

Sutton, R. S. (1995). TD models: Modeling the world as a mixture of time scales. In *Proceedings of the Twelfth International Conference on Machine Learning ICML'95*, (pp. 531–539)., San Mateo, CA. Morgan Kaufmann.

Sutton, R. S. & Barto, A. G. (1998). *Reinforcement Learning. An Introduction*. Cambridge, MA: MIT Press.

Sutton, R. S. & Pinette, B. (1985). The learning of world models by connectionist networks. In *Proceedings of the Seventh Annual Conference of the Cognitive Science Society*, (pp. 54–64).

Watkins, C. J. C. H. (1989). *Learning with Delayed Rewards*. PhD thesis, Cambridge University.